# Almost Linear VC Dimension Bounds for Piecewise Polynomial Networks

**Peter L. Bartlett**
Department of System Engineering
Australian National University
Canberra, ACT 0200
Australia
Peter.Bartlett@anu.edu.au

**Vitaly Maiorov**
Department of Mathematics
Technion, Haifa 32000
Israel

**Ron Meir**
Department of Electrical Engineering
Technion, Haifa 32000
Israel
rmeir@dumbo.technion.ac.il

## Abstract

We compute upper and lower bounds on the VC dimension of feedforward networks of units with piecewise polynomial activation functions. We show that if the number of layers is fixed, then the VC dimension grows as $W \log W$, where $W$ is the number of parameters in the network. This result stands in opposition to the case where the number of layers is unbounded, in which case the VC dimension grows as $W^2$.

## 1  MOTIVATION

The VC dimension is an important measure of the complexity of a class of binary-valued functions, since it characterizes the amount of data required for learning in the PAC setting (see [BEHW89, Vap82]). In this paper, we establish upper and lower bounds on the VC dimension of a specific class of multi-layered feedforward neural networks. Let $\mathcal{F}$ be the class of binary-valued functions computed by a feedforward neural network with $W$ weights and $k$ computational (non-input) units, each with a piecewise polynomial activation function. Goldberg and Jerrum [GJ95] have shown that $\mathrm{VCdim}(\mathcal{F}) \leq c_1(W^2 + Wk) = O(W^2)$, where $c_1$ is a constant. Moreover, Koiran and Sontag [KS97] have demonstrated such a network that has $\mathrm{VCdim}(\mathcal{F}) \geq c_2 W^2 = \Omega(W^2)$, which would lead one to conclude that the bounds

are in fact tight up to a constant. However, the proof used in [KS97] to establish the lower bound made use of the fact that the number of layers can grow with $W$. In practical applications, this number is often a small constant. Thus, the question remains as to whether it is possible to obtain a better bound in the realistic scenario where the number of layers is fixed.

The contribution of this work is the proof of upper and lower bounds on the VC dimension of piecewise polynomial nets. The upper bound behaves as $O(WL^2 + WL \log WL)$, where $L$ is the number of layers. If $L$ is fixed, this is $O(W \log W)$, which is superior to the previous best result which behaves as $O(W^2)$. Moreover, using ideas from [KS97] and [GJ95] we are able to derive a lower bound on the VC dimension which is $\Omega(WL)$ for $L = O(W)$. Maass [Maa94] shows that three-layer networks with threshold activation functions and binary inputs have VC dimension $\Omega(W \log W)$, and Sakurai [Sak93] shows that this is also true for two-layer networks with threshold activation functions and real inputs. It is easy to show that these results imply similar lower bounds if the threshold activation function is replaced by any piecewise polynomial activation function $f$ that has bounded and distinct limits $\lim_{x \to -\infty} f(x)$ and $\lim_{x \to \infty} f(x)$. We thus conclude that if the number of layers $L$ is fixed, the VC dimension of piecewise polynomial networks with $L \geq 2$ layers and real inputs, and of piecewise polynomial networks with $L \geq 3$ layers and binary inputs, grows as $W \log W$. We note that for the piecewise polynomial networks considered in this work, it is easy to show that the VC dimension and pseudo-dimension are closely related (see e.g. [Vid96]), so that similar bounds (with different constants) hold for the pseudo-dimension. Independently, Sakurai has obtained similar upper bounds and improved lower bounds on the VC dimension of piecewise polynomial networks (see [Sak99]).

## 2 UPPER BOUNDS

We begin the technical discussion with precise definitions of the VC-dimension and the class of networks considered in this work.

**Definition 1** *Let $X$ be a set, and $\mathcal{A}$ a system of subsets of $X$. A set $S = \{x_1, \ldots, x_n\}$ is shattered by $\mathcal{A}$ if, for every subset $B \subseteq S$, there exists a set $A \in \mathcal{A}$ such that $S \cap A = B$. The VC-dimension of $\mathcal{A}$, denoted by $\mathrm{VCdim}(\mathcal{A})$, is the largest integer $n$ such that there exists a set of cardinality $n$ that is shattered by $\mathcal{A}$.*

Intuitively, the VC dimension measures the size, $n$, of the largest set of points for which all possible $2^n$ labelings may be achieved by sets $A \in \mathcal{A}$. It is often convenient to talk about the VC dimension of classes of indicator functions $\mathcal{F}$. In this case we simply identify the sets of points $x \in X$ for which $f(x) = 1$ with the subsets of $\mathcal{A}$, and use the notation $\mathrm{VCdim}(\mathcal{F})$.

A feedforward multi-layer network is a directed acyclic graph that represents a parametrized real-valued function of $d$ real inputs. Each node is called either an input unit or a computation unit. The computation units are arranged in $L$ layers. Edges are allowed from input units to computation units. There can also be an edge from a computation unit to another computation unit, but only if the first unit is in a lower layer than the second. There is a single unit in the final layer, called the output unit. Each input unit has an associated real value, which is one of the components of the input vector $x \in \mathbf{R}^d$. Each computation unit has an associated real value, called the unit's output value. Each edge has an associated real parameter, as does each computation unit. The output of a computation unit is given by $\sigma \left( \sum_e w_e z_e + w_0 \right)$, where the sum ranges over the set of edges leading to

the unit, $w_e$ is the parameter (weight) associated with edge $e$, $z_e$ is the output value of the unit from which edge $e$ emerges, $w_0$ is the parameter (bias) associated with the unit, and $\sigma : \mathbf{R} \to \mathbf{R}$ is called the activation function of the unit. The argument of $\sigma$ is called the *net input* of the unit. We suppose that in each unit except the output unit, the activation function is a fixed piecewise polynomial function of the form

$$\sigma(u) = \phi_i(u) \qquad \text{for } u \in [t_{i-1}, t_i),$$

for $i = 1, \ldots, p+1$ (and set $t_0 = -\infty$ and $t_{p+1} = \infty$), where each $\phi_i$ is a polynomial of degree no more than $l$. We say that $\sigma$ has $p$ break-points, and degree $l$. The activation function in the output unit is the identity function. Let $k_i$ denote the number of computational units in layer $i$ and suppose there is a total of $W$ parameters (weights and biases) and $k$ computational units ($k = k_1 + k_2 + \cdots + k_{L-1} + 1$). For input $x$ and parameter vector $a \in A = \mathbf{R}^W$, let $f(x, a)$ denote the output of this network, and let $\mathcal{F} = \{x \mapsto f(x, a) : a \in \mathbf{R}^W\}$ denote the class of functions computed by such an architecture, as we vary the $W$ parameters. We first discuss the computation of the VC dimension, and thus consider the class of functions $\mathrm{sgn}(\mathcal{F}) = \{x \mapsto \mathrm{sgn}(f(x, a)) : a \in \mathbf{R}^W\}$.

Before giving the main theorem of this section, we present the following result, which is a slight improvement of a result due to Warren (see [ABar], Chapter 8).

**Lemma 2.1** *Suppose* $f_1(\cdot), f_2(\cdot), \ldots, f_m(\cdot)$ *are fixed polynomials of degree at most* $l$ *in* $n \leq m$ *variables. Then the number of distinct sign vectors* $\{\mathrm{sgn}(f_1(a)), \ldots, \mathrm{sgn}(f_m(a))\}$ *that can be generated by varying* $a \in \mathbf{R}^n$ *is at most* $2(2eml/n)^n$.

We then have our main result:

**Theorem 2.1** *For any positive integers* $W$, $k \leq W$, $L \leq W$, $l$, *and* $p$, *consider a network with real inputs, up to* $W$ *parameters, up to* $k$ *computational units arranged in* $L$ *layers, a single output unit with the identity activation function, and all other computation units with piecewise polynomial activation functions of degree* $l$ *and with* $p$ *break-points. Let* $\mathcal{F}$ *be the class of real-valued functions computed by this network. Then*

$$\mathrm{VCdim}(\mathrm{sgn}(\mathcal{F})) \leq 2WL\log(2eWLpk) + 2WL^2\log(l+1) + 2L .$$

Since $L$ and $k$ are $O(W)$, for fixed $l$ and $p$ this implies that

$$\mathrm{VCdim}(\mathrm{sgn}(\mathcal{F})) = O(WL\log W + WL^2).$$

Before presenting the proof, we outline the main idea in the construction. For any fixed input $x$, the output of the network $f(x, a)$ corresponds to a piecewise polynomial function in the parameters $a$, of degree no larger than $(l+1)^{L-1}$ (recall that the last layer is linear). Thus, the parameter domain $A = \mathbf{R}^W$ can be split into regions, in each of which the function $f(x, \cdot)$ is polynomial. From Lemma 2.1, it is possible to obtain an upper bound on the number of sign assignments that can be attained by varying the parameters of a set of polynomials. The theorem will be established by combining this bound with a bound on the number of regions.

PROOF OF THEOREM 2.1 For an arbitrary choice of $m$ points $x_1, x_2, \ldots, x_m$, we wish to bound

$$K = |\{(\mathrm{sgn}(f(x_1, a)), \ldots, \mathrm{sgn}(f(x_m, a))) : a \in A\}| .$$

Fix these $m$ points, and consider a partition $\{S_1, S_2, \dots, S_N\}$ of the parameter domain $A$. Clearly

$$K \leq \sum_{i=1}^{N} |\{(\text{sgn}(f(x_1, a)), \dots, \text{sgn}(f(x_m, a))) : a \in S_i\}|.$$

We choose the partition so that within each region $S_i$, $f(x_1, \cdot), \dots, f(x_m, \cdot)$ are all fixed polynomials of degree no more than $(l+1)^{L-1}$. Then, by Lemma 2.1, each term in the sum above is no more than

$$2 \left( \frac{2em(l+1)^{L-1}}{W} \right)^W. \tag{1}$$

The only remaining point is to construct the partition and determine an upper bound on its size. The partition is constructed recursively, using the following procedure. Let $\mathcal{S}_1$ be a partition of $A$ such that, for all $S \in \mathcal{S}_1$, there are constants $b_{h,i,j} \in \{0, 1\}$ for which

$$\text{sgn}(p_{h,x_j}(a) - t_i) = b_{h,i,j} \qquad \text{for all } a \in S,$$

where $j \in \{1, \dots, m\}$, $h \in \{1, \dots, k_1\}$ and $i \in \{1, \dots, p\}$. Here $t_i$ are the breakpoints of the piecewise polynomial activation functions, and $p_{h,x_j}$ is the affine function describing the net input to the $h$-th unit in the first layer, in response to $x_j$. That is,

$$p_{h,x_j} = a_h \cdot x_j + a_{h,0},$$

where $a_h \in \mathbf{R}^d$, $a_{h,0} \in \mathbf{R}$ are the weights of the $h$-th unit in the first layer. Note that the partition $\mathcal{S}_1$ is determined solely by the parameters corresponding to the first hidden layer, as the input to this layer is unaffected by the other parameters. Clearly, for $a \in S$, the output of any first layer unit in response to an $x_j$ is a fixed polynomial in $a$.

Now, let $W_1, \dots, W_L$ be the number of variables used in computing the unit outputs up to layer $1, \dots, L$ respectively (so $W_L = W$), and let $k_1, \dots, k_L$ be the number of computation units in layer $1, \dots, L$ respectively (recall that $k_L = 1$). Then we can choose $\mathcal{S}_1$ so that $|\mathcal{S}_1|$ is no more than the number of sign assignments possible with $mk_1p$ affine functions in $W_1$ variables. Lemma 2.1 shows that $|\mathcal{S}_1| \leq 2 \left( \frac{2emk_1p}{W_1} \right)^{W_1}$.

Now, we define $\mathcal{S}_n$ (for $n > 1$) as follows. Assume that for all $S$ in $\mathcal{S}_{n-1}$ and all $x_j$, the net input of every unit in layer $n$ in response to $x_j$ is a fixed polynomial function of $a \in S$, of degree no more than $(l+1)^{n-1}$. Let $\mathcal{S}_n$ be a partition of $A$ that is a refinement of $\mathcal{S}_{n-1}$ (that is, for all $S \in \mathcal{S}_n$, there is an $S' \in \mathcal{S}_{n-1}$ with $S \subseteq S'$), such that for all $S \in \mathcal{S}_n$ there are constants $b_{h,i,j} \in \{0, 1\}$ such that

$$\text{sgn}(p_{h,x_j}(a) - t_i) = b_{h,i,j} \qquad \text{for all } a \in S, \tag{2}$$

where $p_{h,x_j}$ is the polynomial function describing the net input of the $h$-th unit in the $n$-th layer, in response to $x_j$, when $a \in S$. Since $S \subseteq S'$ for some $S' \in \mathcal{S}_{n-1}$, (2) implies that the output of each $n$-th layer unit in response to an $x_j$ is a fixed polynomial in $a$ of degree no more than $l(l+1)^{n-1}$, for all $a \in S$.

Finally, we can choose $\mathcal{S}_n$ such that, for all $S' \in \mathcal{S}_{n-1}$ we have $|\{S \in \mathcal{S}_n : S \subseteq S'\}|$ is no more than the number of sign assignments of $mk_np$ polynomials in $W_n$ variables of degree no more than $(l+1)^{n-1}$, and by Lemma 2.1 this is no more than $2 \left( \frac{2emk_np(l+1)^{n-1}}{W_n} \right)^{W_n}$. Notice also that the net input of every unit in layer $n+1$ in

response to $x_j$ is a fixed polynomial function of $a \in S \in \mathcal{S}_n$ of degree no more than $(l+1)^n$.

Proceeding in this way we get a partition $\mathcal{S}_{L-1}$ of $A$ such that for $S \in \mathcal{S}_{L-1}$ the network output in response to any $x_j$ is a fixed polynomial of $a \in S$ of degree no more than $l(l+1)^{L-2}$. Furthermore,

$$
\begin{aligned}
|\mathcal{S}_{L-1}| &\leq 2 \left( \frac{2emk_1p}{W_1} \right)^{W_1} \prod_{i=2}^{L-1} 2 \left( \frac{2emk_ip(l+1)^{i-1}}{W_i} \right)^{W_i} \\
&\leq \prod_{i=1}^{L-1} 2 \left( \frac{2emk_ip(l+1)^{i-1}}{W_i} \right)^{W_i}.
\end{aligned}
$$

Multiplying by the bound (1) gives the result

$$
K \leq \prod_{i=1}^{L} 2 \left( \frac{2emk_ip(l+1)^{i-1}}{W_i} \right)^{W_i}.
$$

Since the points $x_1, \ldots, x_m$ were chosen arbitrarily, this gives a bound on the maximal number of dichotomies induced by $a \in A$ on $m$ points. An upper bound on the VC-dimension is then obtained by computing the largest value of $m$ for which this number is at least $2^m$, yielding

$$
\begin{aligned}
m &< L + \sum_{i=1}^{L} W_i \log \left( \frac{2empk_i(l+1)^{i-1}}{W_i} \right) \\
&< L\left[1 + (L-1)W \log(l+1) + W \log(2empk)\right],
\end{aligned}
$$

where all logarithms are to the base 2. We conclude (see for example [Vid96] Lemma 4.4) that

$$
\mathrm{VCdim}(\mathcal{F}) \leq 2L\left[(L-1)W \log(l+1) + W \log\left(2eWLpk\right) + 1\right].
$$

∎

We briefly mention the application of this result to the problem of learning a regression function $\mathbf{E}[Y|X = x]$, from $n$ input/output pairs $\{(X_i, Y_i)\}_{i=1}^n$, drawn independently at random from an unknown distribution $P(X, Y)$. In the case of quadratic loss, $L(f) = \mathbf{E}(Y - f(X))^2$, one can show that there exist constants $c_1 \geq 1$ and $c_2$ such that

$$
\mathbf{E}L(\hat{f}_n) \leq s^2 + c_1 \inf_{f \in \mathcal{F}} \tilde{L}(f) + c_2 \frac{M\mathrm{Pdim}(\mathcal{F}) \log n}{n},
$$

where $s^2 = \mathbf{E}\left[Y - \mathbf{E}[Y|X]\right]^2$ is the noise variance, $\tilde{L}(f) = \mathbf{E}\left[(\mathbf{E}[Y|X] - f(X))^2\right]$ is the approximation error of $f$, and $\hat{f}_n$ is a function from the class $\mathcal{F}$ that approximately minimizes the sample average of the quadratic loss. Making use of recently derived bounds [MM97] on the approximation error, $\inf_{f \in \mathcal{F}} \tilde{L}(f)$, which are equal, up to logarithmic factors, to those obtained for networks of units with the standard sigmoidal function $\sigma(u) = (1 + e^{-u})^{-1}$, and combining with the considerably lower pseudo-dimension bounds for piecewise polynomial networks, we obtain much better error rates than are currently available for sigmoid networks.

## 3   LOWER BOUND

We now compute a lower bound on the VC dimension of neural networks with continuous activation functions. This result generalizes the lower bound in [KS97], since it holds for any number of layers.

**Theorem 3.1** *Suppose $f : \mathbf{R} \to \mathbf{R}$ has the following properties:*

*1. $\lim_{\alpha \to \infty} f(\alpha) = 1$ and $\lim_{\alpha \to -\infty} f(\alpha) = 0$, and*

*2. $f$ is differentiable at some point $x_0$ with derivative $f'(x_0) \neq 0$.*

*Then for any $L \geq 1$ and $W \geq 10L - 14$, there is a feedforward network with the following properties: The network has $L$ layers and $W$ parameters, the output unit is a linear unit, all other computation units have activation function $f$, and the set $\mathrm{sgn}(\mathcal{F})$ of functions computed by the network has*

$$\mathrm{VCdim}(\mathrm{sgn}(\mathcal{F})) \geq \left\lfloor \frac{L}{2} \right\rfloor \left\lfloor \frac{W}{2} \right\rfloor,$$

*where $\lfloor u \rfloor$ is the largest integer less than or equal to $u$.*

PROOF    As in [KS97], the proof follows that of Theorem 2.5 in [GJ95], but we show how the functions described in [GJ95] can be computed by a network, and keep track of the number of parameters and layers required. We first prove the lower bound for a network containing linear threshold units and linear units (with the identity activation function), and then show that all except the output unit can be replaced by units with activation function $f$, and the resulting network still shatters the same set. For further details of the proof, see the full paper [BMM98].

Fix positive integers $M, N \in \mathbf{N}$. We now construct a set of $MN$ points, which may be shattered by a network with $O(N)$ weights and $O(M)$ layers. Let $\{a_i\}$, $i = 1, 2, \ldots, N$ denote a set of $N$ parameters, where each $a_i \in [0, 1)$ has an $M$-bit binary representation $a_i = \sum_{j=1}^{M} 2^{-j} a_{i,j}$, $a_{i,j} \in \{0, 1\}$, i.e. the $M$-bit base two representation of $a_i$ is $a_i = 0.a_{i,1} a_{i,2} \ldots a_{i,M}$. We will consider inputs in $B_N \times B_M$, where $B_N = \{e_i : 1 \leq i \leq N\}$, $e_i \in \{0, 1\}^N$ has $i$-th bit 1 and all other bits 0, and $B_M$ is defined similarly. We show how to extract the bits of the $a_i$, so that for input $x = (e_l, e_m)$ the network outputs $a_{l,m}$. Since there are $NM$ inputs of the form $(e_l, e_m)$, and $a_{l,m}$ can take on all possible $2^{MN}$ values, the result will follow. There are three stages to the computation of $a_{l,m}$: (1) computing $a_l$, (2) extracting $a_{l,k}$ from $a_l$, for every $k$, and (3) selecting $a_{l,m}$ among the $a_{l,k}$s.

Suppose the network input is $x = ((u_1, \ldots, u_N), (v_1, \ldots, v_M)) = (e_l, e_m)$. Using one linear unit we can compute $\sum_{i=1}^{N} u_i a_i = a_l$. This involves $N + 1$ parameters and one computation unit in one layer. In fact, we only need $N$ parameters, but we need the extra parameter when we show that this linear unit can be replaced by a unit with activation function $f$.

Consider the parameter $c_k = 0.a_{l,k} \ldots a_{l,M}$, that is, $c_k = \sum_{j=k}^{M} 2^{k-1-j} a_{l,j}$ for $k = 1, \ldots, M$. Since $c_k \geq 1/2$ iff $a_{l,k} = 1$, clearly $\mathrm{sgn}(c_k - 1/2) = a_{l,k}$ for all $k$. Also, $c_1 = a_l$ and $c_k = 2c_{k-1} - a_{l,k-1}$. Thus, consider the recursion

$$c_k = 2c_{k-1} - a_{l,k-1}$$
$$a_{l,k} = \mathrm{sgn}(c_k - 1/2),$$

with initial conditions $c_1 = a_l$ and $a_{l,1} = \mathrm{sgn}(a_l - 1/2)$. Clearly, we can compute $a_{l,1}, \ldots, a_{l,M-1}$ and $c_2, \ldots, c_{M-1}$ in another $2(M-2)+1$ layers, using $5(M-2)+2$ parameters in $2(M-2)+1$ computational units.

We could compute $a_{l,M}$ in the same way, but the following approach gives fewer layers. Set $b = \mathrm{sgn}\left(2c_{M-1} - a_{l,M-1} - \sum_{i=1}^{M-1} v_i\right)$. If $m \neq M$ then $b = 0$. If $m = M$ then the input vector $(v_1, \ldots, v_M) = e_M$, and thus $\sum_{i=1}^{M-1} v_i = 0$, implying that $b = \mathrm{sgn}(c_M) = \mathrm{sgn}(0.a_{l,M}) = a_{l,M}$.

In order to conclude the proof, we need to show how the variables $a_{l,m}$ may be recovered, depending on the inputs $(v_1, v_2, \ldots, v_M)$. We then have $a_{l,m} = b \vee \bigvee_{i=1}^{M-1}(a_{l,i} \wedge v_i)$. Since for boolean $x$ and $y$, $x \wedge y = \text{sgn}(x + y - 3/2)$, and $\bigvee_{i=1}^{M} x_i = \text{sgn}(\sum_{i=1}^{M} x_i - 1/2)$, we see that the computation of $a_{l,m}$ involves an additional $5M$ parameters in $M + 1$ computational units, and adds another 2 layers.

In total, there are $2M$ layers and $10M + N - 7$ parameters, and the network shatters a set of size $NM$. Clearly, we can add parameters and layers without affecting the function of the network. So for any $L, W \in \mathbf{N}$, we can set $M = \lfloor L/2 \rfloor$ and $N = W + 7 - 10M$, which is at least $\lfloor W/2 \rfloor$ provided $W \geq 10L - 14$. In that case, the VC-dimension is at least $\lfloor L/2 \rfloor \lfloor W/2 \rfloor$.

The network just constructed uses linear threshold units and linear units. However, it is easy to show (see [KS97], Theorem 5) that each unit except the output unit can be replaced by a unit with activation function $f$ so that the network still shatters the set of size $MN$. For linear units, the input and output weights are scaled so that the linear function can be approximated to sufficient accuracy by $f$ in the neighborhood of the point $x_0$. For linear threshold units, the input weights are scaled so that the behavior of $f$ at infinity accurately approximates a linear threshold function. ∎

# References

[ABar]     M. Anthony and P. L. Bartlett. *Neural Network Learning: Theoretical Foundations.* Cambridge University Press, 1999 (to appear).

[BEHW89]     A. Blumer, A. Ehrenfeucht, D. Haussler, and M. K. Warmuth. Learnability and the Vapnik-Chervonenkis dimension. *J. ACM*, 36(4):929–965, 1989.

[BMM98]     P. L. Bartlett, V. Maiorov, and R. Meir. Almost linear VC-dimension bounds for piecewise polynomial networks. *Neural Computation*, 10:2159–2173, 1998.

[GJ95]     P.W. Goldberg and M.R. Jerrum. Bounding the VC Dimension of Concept Classes Parameterized by Real Numbers. *Machine Learning*, 18:131–148, 1995.

[KS97]     P. Koiran and E.D. Sontag. Neural Networks with Quadratic VC Dimension. *Journal of Computer and System Science*, 54:190–198, 1997.

[Maa94]     W. Maass. Neural nets with superlinear VC-dimension. *Neural Computation*, 6(5):877–884, 1994.

[MM97]     V. Maiorov and R. Meir. On the Near Optimality of the Stochastic Approximation of Smooth Functions by Neural Networks. Submitted for publication, 1997.

[Sak93]     A. Sakurai. Tighter bounds on the VC-dimension of three-layer networks. In *World Congress on Neural Networks*, volume 3, pages 540–543, Hillsdale, NJ, 1993. Erlbaum.

[Sak99]     A. Sakurai. Tight bounds for the VC-dimension of piecewise polynomial networks. In *Advances in Neural Information Processing Systems*, volume 11. MIT Press, 1999.

[Vap82]     V. N. Vapnik. *Estimation of Dependences Based on Empirical Data.* Springer-Verlag, New York, 1982.

[Vid96]     M Vidyasagar. *A Theory of Learning and Generalization.* Springer Verlag, New York, 1996.
